# Representation and Induction of Finite State Machines using Time-Delay Neural Networks

**Daniel S. Clouse**
Computer Science & Engineering Dept.
University of California, San Diego
La Jolla, CA 92093-0114
dclouse@ucsd.edu

**C. Lee Giles**
NEC Research Institute
4 Independence Way
Princeton, NJ 08540
giles@research.nj.nec.com

**Bill G. Horne**
NEC Research Institute
4 Independence Way
Princeton, NJ 08540
horne@research.nj.nec.com

**Garrison W. Cottrell**
Computer Science & Engineering Dept.
University of California, San Diego
La Jolla, CA 92093-0114
gcottrell@ucsd.edu

## Abstract

This work investigates the representational and inductive capabilities of *time-delay neural networks* (TDNNs) in general, and of two subclasses of TDNN, those with delays only on the inputs (IDNN), and those which include delays on hidden units (HDNN). Both architectures are capable of representing the same class of languages, the *definite memory machine* (DMM) languages, but the delays on the hidden units in the HDNN helps it outperform the IDNN on problems composed of repeated features over short time windows.

## 1 Introduction

In this paper we consider the representational and inductive capabilities of *time-delay neural networks* (TDNN) [Waibel et al., 1989] [Lang et al., 1990], also known as NNFIR [Wan, 1993]. A TDNN is a feed-forward network in which the set of inputs to any node $i$ may include the output from previous layers not only in the current time step $t$, but from $d$ earlier time steps as well. The activation function

for node $i$ at time $t$ in such a network is given by equation 1:

$$y_i^t = h(\sum_{j=1}^{i-1} \sum_{k=0}^{d} y_j^{t-k} w_{ijk}) \tag{1}$$

where $y_i^t$ is the activation of node $i$ at time $t$, $w_{ijk}$ is the connection strength from node $j$ to node $i$ at delay $k$, and $h$ is the squashing function.

TDNNs have been used in speech recognition [Waibel et al., 1989], and time series prediction [Wan, 1993]. In this paper we concentrate on the language induction problem. A training set of variable-length strings taken from a discrete alphabet $\{0, 1\}$ is generated. Each string is labeled as to whether it is in some language L or not. The network must learn to discriminate strings which are in the language from those which are not, not only for the training set strings, but for strings the network has never seen before. The language induction problem provides a simple, familiar domain in which to gain insight into the capabilities of different network architectures.

Specifically, in this paper, we will look at the representational and inductive capabilities of the general class of TDNNs versus a subclass of TDNNs, the *input-delay neural networks* (IDNNs). An IDNN is a TDNN in which delays are limited to the network inputs. In section 2, we will show that the classes of functions representable by general TDNNs and IDNNs are equivalent. In section 3, we will show that the class of languages representable by the TDNNs, are the *definite memory machine* (DMM) languages. In section 4, we will demonstrate the inductive capability of the TDNNs in a simulation in which a large DMM is learned using a small percentage of the possible, short training examples. In section 5, a second set of simulations will show the difference between *representational* and *inductive bias*, and will demonstrate the utility of internal delays in a TDNN network.

## 2    TDNNs and IDNNs Are Functionally Equivalent

Since every IDNN is also a TDNN, the set of functions computable by any TDNN includes all those computable by the IDNNs. [Wan, 1993] also shows that the IDNNs can compute any function computable by the TDNNs making these two classes of network architectures functionally equivalent. For completeness, here we include a description of how to construct from a TDNN, an equivalent IDNN.

Figure 1a shows a TDNN with a single input $u$ at the current time $(u_t)$, and at four earlier time steps $(u_{t-1} \ldots u_{t-4})$. The inputs to node $R$ consist of the outputs of nodes $P$ and $Q$ at the current time step along with one or two previous time steps. At time $t$, node $P$ computes $f_P(u_t, \ldots u_{t-4})$, a function of the current input and four delays. At time $t-1$, node $P$ computes $f_P(u_{t-1}, \ldots u_{t-5})$. This serves as one of the delayed inputs to node R. This value could also be computed by sliding node $P$ over one step in the input tap-delay line along with its incoming weights as shown in figure 1b. Using this construction, all the internal delays can be removed, and replaced by copies of the original nodes $P$ and $Q$, along with their incoming weights. This method can be applied recursively to remove any internal delay in any TDNN network. Thus, for any function computable by a TDNN, we can construct an IDNN which computes the same function.

## 3    TDNNs Can Represent the DMM Languages

In this section, we show that the set of languages which are representable by some TDNN are exactly those languages representable by the *definite memory machines*

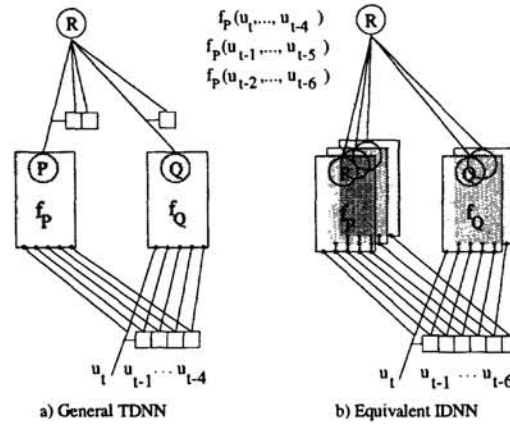

Figure 1: Constructing an IDNN equivalent to a given TDNN

(DMMs). According to Kohavi (1978) a DMM of order $d$ is a *finite state machine* (FSM) whose present state can always be determined uniquely from the knowledge of the most recent $d$ inputs. We equivalently define a DMM of order $d$ as an FSM whose accepting/rejecting behavior is a function of only the most recent $d$ inputs.

To fit TDNNs and IDNNs into the language induction framework, we consider only networks with a single 0/1 input. Since any boolean function can be represented by a feed-forward network with enough hidden units [Horne and Hush, 1994], an IDNN exists which can perform the mapping from $d$ most recent inputs to any accepting/rejecting behavior. Therefore, any DMM language can be represented by some IDNN. Since every IDNN computes a function of its most recent $d$ inputs, by the definition of DMM, there is no boolean output IDNN which represents a non-DMM language. Therefore, the IDNNs represent exactly the DMM languages. Since the TDNN and IDNN classes are functionally equivalent, TDNNs implement exactly the DMM languages as well.

The shift register behavior of the input tap-delay line in an IDNN completely determines the state transition behavior of any machine represented by the network. This state transition behavior is fixed by the architecture. For example, figure 2a shows the state transition diagram for any machine representable by an IDNN with two input delays. The mapping from the current state to "accept" or "reject" is all that can be changed with training. Clouse et al. (1994) describes the conditions under which such a mapping results in a minimal FSM. All mappings used in the subsequent simulations are minimal FSM mappings.

## 4 Simulation 1: Large DMM

To demonstrate the close relationship between TDNNs and DMMs, here we present the results of a simulation in which we trained an IDNN to reproduce the behavior of a DMM of order 11. The mapping function for the DMM is given in equation 2. Figure 2b shows the minimal 2048 state transition diagram required to represent the DMM. The symbol $\leftrightarrow$ in equation 2 represents the *if-and-only-if* function. The overbar notation, $\overline{u}_k$, represents the negation of $u_k$, the input at time $k$. $Y_k$ is the network output at time $k$. $Y_k > 0.5$ is interpreted as "accept the string seen so far." $Y_k \leq 0.5$ means "reject."

$$y_k = u_{k-10} \leftrightarrow (\overline{u}_k \overline{u}_{k-1} \overline{u}_{k-2} + \overline{u}_{k-2} u_{k-3} + u_{k-1} u_{k-2}) \tag{2}$$

To create training and test sets, we randomly split in two the set of all 4094

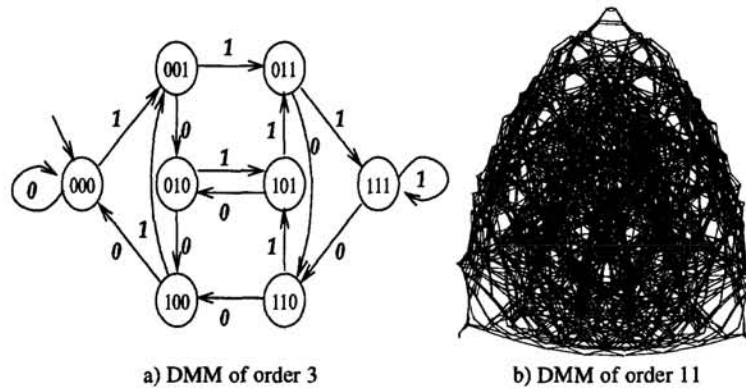

a) DMM of order 3                    b) DMM of order 11

Figure 2: Transition diagrams for two DMMs.

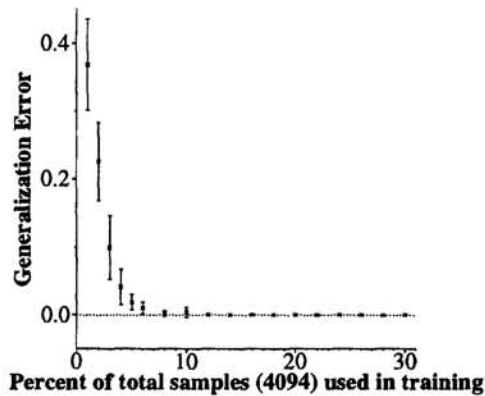

Figure 3: Generalization error on 2048 state DMM.

strings of length 11 or less. We will report results using various percentages of possible strings for the training set. The IDNN had 10 input tap-delays, and seven hidden units. All tap-delays were cleared to 0 before introduction of a new input string. Weights were trained using online back propagation with learning rate 0.25, and momentum 0.25. To speed up the algorithm, weights were updated only if the absolute error on an example was greater than 0.2. Training was stopped when weight updates were required for no examples in the training set. This generally required 200 epochs or fewer, though there were trials which required almost 4000 epochs.

Each point in figure 3 represents the mean classification error on the test set across 20 trials. Error bars indicate one standard deviation on each side of the mean. Each trial consists of a different randomly-chosen training set. The graph plots error at various training set sizes. Note that with training sets as small as 12 percent of possible strings the network generalizes perfectly to the remaining 88 percent. This kind of performance is possible because of the close match between the representational bias of the IDNN and this specific problem.

## 5   Simulation 2: Inductive biases of IDNNs and HDNNs

In section 2, we showed that the IDNNs and general TDNNs can *represent* the same class of functions. It does not follow that these two architectures are equally capable of *learning* the same functions. In this section, we show that the inductive biases are

indeed different. We will present our intuitions about the kinds of problems each architecture is well suited to learning, then back up our intuitions with supporting simulations.

In the following simulations, we compare two specific networks. The network representing the general TDNNs includes delays on hidden layer outputs. We'll refer to this as the *hidden delay neural network* or HDNN. All delays in the second network are confined to the network inputs, and so we call this the IDNN.

We have been careful to design the two networks to be comparable in size. Each of the networks contains two hidden layers. The first hidden layer of the IDNN has four units, and the second five. The IDNN has eight input delays. Each of the two hidden layers of the HDNN has three units. The HDNN has three input delays, and five delays on the output of each node of the first hidden layer. Note that in each network the longest path from input to output requires eight delays. The number of weights, including bias weights, are also similar – 76 for the HDNN, and 79 for the IDNN.

In order for the size of the two networks to be similar, the HDNN must have fewer delays on the network inputs. If we think of each unit in the first hidden layer as a feature detector, the feature detectors in the HDNN will span a smaller time window than the IDNN. On the other hand, the HDNN has a second set of delays which saves the output of the feature detectors over several time steps. If some narrow feature repeats over time, this second set of delays should help the HDNN to pick up this regularity. The IDNN, lacking the internal delays, should find it more difficult to detect this kind of repeated regularity.

To test these ideas, we generated four DMM problems. We call equation 3 the *narrow-repeated* problem because it contains a number of identical terms shifted in time, and because each of these terms is narrow enough to fit in the time window of the HDNN first layer feature detectors.

$$y_k = u_{k-8} \leftrightarrow (u_k u_{k-2} \overline{u}_{k-3} + u_{k-1} u_{k-3} \overline{u}_{k-4} + u_{k-3} u_{k-5} \overline{u}_{k-6} + u_{k-4} u_{k-6} \overline{u}_{k-7})$$
(3)

The *wide-repeated* problem, represented by equation 4, is identical to the narrow-repeated problem except that each term has been stretched so that it will no longer fit in the HDNN feature detector time window.

$$y_k = u_{k-8} \leftrightarrow (u_k u_{k-2} \overline{u}_{k-4} + u_{k-1} u_{k-3} \overline{u}_{k-5} + u_{k-2} u_{k-4} \overline{u}_{k-6} + u_{k-3} u_{k-5} \overline{u}_{k-7})$$
(4)

The *narrow-unrepeated* problem, represented by equation 5, is composed of narrow terms, but none of these terms is simply a shifted reproduction of another.

$$y_k = u_{k-8} \leftrightarrow (u_k u_{k-2} \overline{u}_{k-3} + \overline{u}_{k-1} u_{k-3} u_{k-4} + u_{k-3} \overline{u}_{k-5} u_{k-6} + \overline{u}_{k-4} \overline{u}_{k-6} \overline{u}_{k-7})$$
(5)

Lastly, the *wide-unrepeated* problem of equation 6 contains wide terms which do not repeat.

$$y_k = u_{k-8} \leftrightarrow (u_k u_{k-3} \overline{u}_{k-4} + \overline{u}_{k-1} u_{k-4} u_{k-5} + u_{k-2} \overline{u}_{k-5} u_{k-6} + \overline{u}_{k-3} \overline{u}_{k-6} \overline{u}_{k-7})$$
(6)

Each problem in this section requires a minimum of 512 states to represent.

Similar to the simulation of section 3, we trained both networks on subsets of all possible strings of length 9 or less. Since these problems were more difficult than that of section 3, often the networks were unable to find a solution which performed perfectly on the training set. In this case, training was stopped after 8000 epochs. The results reported later include these trials as well as trials in which training ended because of perfect performance on the training set. Training for the HDNN

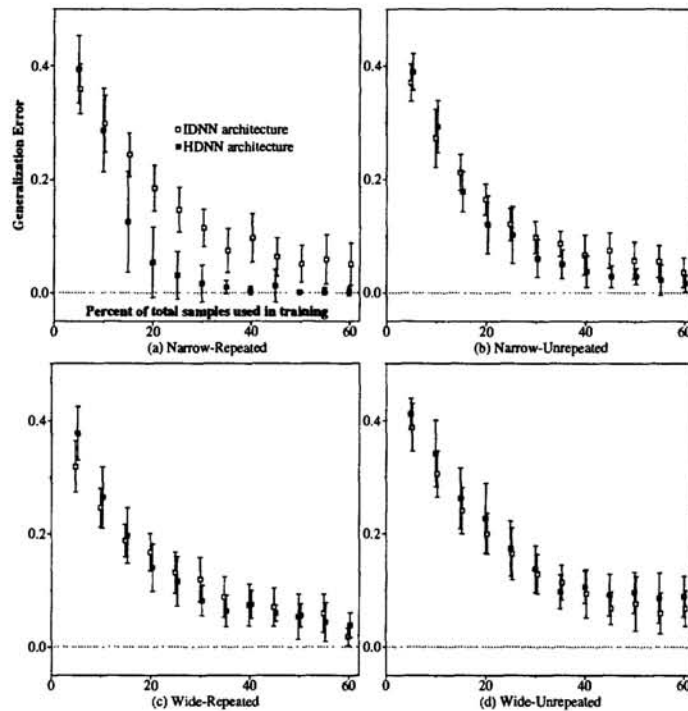

Figure 4: Generalization of a HDNN and an IDNN on four DMM problems

was identical to that of the IDNN except that error was propagated back across the internal delays as in Wan (1993).

Figure 4 plots generalization error versus percentage of possible strings used in training for the two networks for each of the four DMM problems. If our intuitions were correct we would expect to see evidence here that the effect of wider terms, and lack of repetition would have a stronger adverse effect on the HDNN network than on the IDNN. This is exactly what we see. The position of the curve for the IDNN network is stable compared to that of the HDNN when changes are made to the width and repetition factors.

Statistical analysis supports this conclusion. We ran an ANOVA test [Rice, 1988] with four factors (which network, term width, term repetition, and training set size) on the data summarized by the graphs of figure 4. The test detected a significant interaction between the network and width factors ($MS_{net \times wid} = 0.3430$, $F(1, 1824) = 234.4$), and between the network and repetition factors ($MS_{net \times rep} = 0.1181$, $F(1, 1824) = 80.694$). These two interactions are significant at $p < 0.001$, agreeing with our conclusion that width and repetition each has a stronger effect on the performance of the HDNN network.

Further planned tests reveal that the effects of width and repetition are strong enough to change which network generalizes better. We ran a one-way ANOVA test on each problem individually to see which network performs better across the entire curve. The tests reveal that the HDNN performs with significantly less error than the IDNN in the narrow-repeated problem ($MS_{error} = 0.0015$, $MS_{net} = 0.5400$, $F(1, 1824) = 369.0$), and in the narrow-unrepeated problem ($MS_{net} = 0.0683$, $F(1, 1824) = 46.7$). Performance of the IDNN is significantly better in the wide-unrepeated problem ($MS_{net} = 0.0378$, $F(1, 1824) = 25.83$). All of these comparisons are significant at $p < 0.001$. The test on the wide-repeated problem finds no significant difference in performance of the two networks ($MS_{net} = 0.0004$,

$F(1, 1824) = 0.273, p > 0.05$).

In addition to confirming our intuitions about the kinds of problems that internal delays should be helpful in solving, this set of simulations demonstrates the difference between representational and inductive bias. For all DMM problems except for the wide-unrepeated one, we were able to find, for each network, at least one set of weights which solve the problem perfectly. Despite the fact that the two networks are both capable of representing the problems, the differing way in which they respond to the width and repetition factors demonstrates a difference in their learning biases.

## 6 Conclusions

This paper presents a number of interesting ideas concerning TDNNs using both theoretical and empirical techniques. On the theoretical side, we have precisely defined the subclass of FSMs which can be represented by TDNNs, the DMM languages. It is interesting to note that this network architecture which has no recurrent connections is capable of representing languages whose transition diagrams require loops.

Other ideas were demonstrated using empirical techniques. First, we have shown that the number of states required to represent an FSM may be a poor predictor of how difficult the language is to learn. We were able to learn a 2048-state FSM using a small percentage of the possible training examples. This is possible because of the close match between the representational bias of the network, and the language learned.

Second, we presented a set of simulations which demonstrated the utility of internal delays in a TDNN. These delays were shown to improve generalization on problems composed of features over short time intervals which reappear repeatedly.

Third, that same set of simulations highlights the difference between representational bias, and inductive bias. Though these two terms are sometimes used interchangeably in the theoretical literature, this work shows that the two concepts are, in fact, separable.

## References

[Clouse et al., 1994] Clouse, D. S., Giles, C. L., Horne, B. G., and Cottrell, G. W. (1994). Learning large debruijn automata with feed-forward neural networks. Technical Report CS94-398, University of California, San Diego, Computer Science and Engineering Dept.

[Horne and Hush, 1994] Horne, B. G. and Hush, D. R. (1994). On the node complexity of neural networks. *Neural Networks*, 7(9):1413–1426.

[Kohavi, 1978] Kohavi, Z. (1978). *Switching and Finite Automata Theory*. McGraw-Hill, Inc., New York, NY, second edition.

[Lang et al., 1990] Lang, K., Waibel, A., and Hinton, G. (1990). A time-delay neural network architecture for isolated word recognition. *Neural Networks*, 3(1):23–44.

[Rice, 1988] Rice, J. A. (1988). *Mathematical Statistics and Data Analysis*. Brooks/Cole Publishing Company, Monterey, California.

[Waibel et al., 1989] Waibel, A., Hanazawa, T., Hinton, G., Shikano, K., and Lang, K. (1989). Phoneme recognition using time–delay neural networks. *IEEE Transactions on Acoustics, Speech and Signal Processing*, 37(3):328–339.

[Wan, 1993] Wan, E. A. (1993). Time series prediction by using a connectionist network with internal delay lines. In Weigend, A. S. and Gershenfeld, N. A., editors, *Time Series Prediction: Forecasting the Future and Understanding the Past*. Addison Wesley.
